# Cholinergic Modulation Preserves Spike Timing Under Physiologically Realistic Fluctuating Input

**Akaysha C. Tang**
The Salk Institute
Howard Hughes Medical Institute
Computational Neurobiology Laboratory
La Jolla, CA 92037

**Andreas M. Bartels**
Zoological Institute
University of Zürich
Zürich
Switzerland

**Terrence J. Sejnowski**
The Salk Institute
Howard Hughes Medical Institute
Computational Neurobiology Laboratory
La Jolla, CA 92037

## Abstract

Neuromodulation can change not only the mean firing rate of a neuron, but also its pattern of firing. Therefore, a reliable neural coding scheme, whether a rate coding or a spike time based coding, must be robust in a dynamic neuromodulatory environment. The common observation that cholinergic modulation leads to a reduction in spike frequency adaptation implies a modification of spike timing, which would make a neural code based on precise spike timing difficult to maintain. In this paper, the effects of cholinergic modulation were studied to test the hypothesis that precise spike timing can serve as a reliable neural code. Using the whole cell patch-clamp technique in rat neocortical slice preparation and compartmental modeling techniques, we show that cholinergic modulation, surprisingly, preserved spike timing in response to a fluctuating inputs that resembles *in vivo* conditions. This result suggests that in vivo spike timing may be much more resistant to changes in neuromodulator concentrations than previous physiological studies have implied.

# 1  Introduction

Recently, there has been a vigorous debate concerning the nature of neural coding (Rieke *et al.* 1996; Stevens and Zador 1995; Shadlen and Newsome 1994). The prevailing view has been that the mean firing rate conveys all information about the sensory stimulus in a spike train and the precise timing of the individual spikes is noise. This belief is, in part, based on a lack of correlation between the precise timing of the spikes and the sensory qualities of the stimulus under study, particularly, on a lack of spike timing repeatability when identical stimulation is delivered. This view has been challenged by a number of recent studies, in which highly repeatable temporal patterns of spikes can be observed both *in vivo* (Bair and Koch 1996; Abeles *et al.* 1993) and *in vitro* (Mainen and Sejnowski 1994). Furthermore, application of information theory to the coding problem in the frog and house fly (Bialek *et al.* 1991; Bialek and Rieke 1992) suggested that additional information could be extracted from spike timing. In the absence of direct evidence for a timing code in the cerebral cortex, the role of spike timing in neural coding remains controversial.

## 1.1  A necessary condition for a spike timing code

If spike timing is important in defining a stimulus, precisely timed spikes must be maintained under a range of physiological conditions. One important aspect of a neuron's environment is the presence of various neuromodulators. Due to their widespread projections in the nervous system, major neuromodulators, such as acetylcholine (ACh) and norepinephrine (NA), can have a profound influence on the firing properties of most neurons. If a change in concentration of a neuromodulator completely alters the temporal structure of the spike train, it would be unlikely that spike timing could serve as a reliable neural code. A major effect of cholinergic modulation on cortical neurons is a reduction in spike frequency adaptation, which is characterized by a shortening of inter-spike-intervals and an increase in neuronal excitability (McCormick 1993; Nicoll 1988). One obvious consequence of this cholinergic effect is a modification of spike timing (Fig. 1A). This modification of spike timing due to a change in neuromodulator concentration would seem to preclude the possibility of a neural code based on precise spike timing.

## 1.2  Re-examination of the cholinergic modulation of spike timing

Despite its popularity, the square pulse stimulus used in most eletrophysiological studies is rarely encountered by a cortical neuron under physiological conditions. The corresponding behavior of the neuron at the input/output level may have limited relevance to the behavior of the neuron under its natural condition, which is characterized *in vivo* by highly fluctuating synaptic inputs. In this paper, we re-examine the effect of cholinergic modulation on spike timing under two contrasting stimulus conditions: the physiologically unrealistic square pulse input versus the more plausible fluctuating input. We report that under physiologically more realistic fluctuating inputs, effects of cholinergic modulation preserved the timing of each individual spike (Fig. 1B). This result is consistent with the hypothesis that spike timing may be relevant to information encoding.

## 2  Methods

### 2.1  Experimental

Using the whole cell patch-clamp technique, we made somatic recordings from layer 2/3 neocortical neurons in the rat visual cortex. Coronal slices of 400 $\mu$m were prepared from 14 to 18 days old Long Evans rats (for details see (Mainen and Sejnowski 1994). Spike trains elicited by current injection of 900 ms were recorded for the square pulse inputs and fluctuating inputs with equal mean synaptic inputs, in the absence and presence of a cholinergic agonist carbachol. The fluctuating inputs were constructed from Gaussian noise and convolved with an alpha function with a time constant of 3 ms, reflecting the time course of the synaptic events. The amplitude of fluctuation was such that the subthreshold membrane potential fluctuation observed in our experiments were comparable to that in whole-cell patch clamp study in vivo (Ferster and Jagadeesh 1992). The cholinergic agonist carbachol at concentrations of 5, 7.5, 15, 30 $\mu$M was delivered through bath perfusion (perfusion time: between 1 and 6 min). For each cell, three sets of blocks were recorded before, during and after carbachol perfusion at a given concentration. Each block contained 20 trials of stimulation under identical experimental conditions.

### 2.2  Simulation

We used a compartmental model of a neocortical neuron to explore the contribution of three potassium conductances affected by cholinergic modulation (Madison *et al.* 1987). Simulations were performed in a reduced 9 compartment model, based on a layer 2 pyramidal cell reconstruction using the NEURON program. The model had five conductances: $g_{Na}$, $g_{K_\nu}$, $g_{K_M}$, $g_{Ca}$, $g_{K(Ca)}$. Membrane resistivity was $40K\Omega cm^2$, capacitance was $1\mu F/\mu m^2$, and axial resistance was $200\Omega cm$. Intrinsic noise was simulated by injecting a randomly fluctuating current to fit the spike jitter observed experimentally. Different potassium conductances were manipulated as independent variables and the spike timing displacement was measured for multiple levels of conductance change corresponding to multiple concentrations of carbachol.

### 2.3  Data analysis

For both experimental and simulation data, first derivatives were used to detect spikes and to determine the timing for spike initiation. Raster plots of the spike trains were derived from the series of membrane potentials for each trial, and a smoothed histogram was then constructed to reflect the instantaneous firing rate for each block of trials under identical stimulation and pharmacological conditions. An event was then defined as a period of increase in instantaneous firing rate that is greater than a threshold level (set at 3 times of the mean firing rate within the block of trials) (Mainen and Sejnowski 1994).

The effect of carbachol on spike timing under fluctuating inputs was quantified by defining the displacement in spike timing for each event, $d_i$, as the time difference between the nearest peaks of the events under carbachol and control condition. The weight for each event, $w_i$, is determined by the peak of the event. The higher the peak, the less the spike jitter. The mean displacement is

$$D = \sum d_i w_i / \sum w_i, \tag{1}$$

where i= 1, 2, ...nth event in the control condition.

## 3    Results

### 3.1    Experimental

The effects of carbachol on spike timing under the square pulse and fluctuating inputs are shown in Fig. 1A and B respectively. In the absence of carbachol, a square pulse input produced a spike train with clear spike frequency adaptation (Fig. 1A1). Similar to previous reports from the literature, addition of carbachol to the perfusion medium reduced spike frequency adaptation (Fig. 1A2). This reduction in spike frequency adaptation is reflected in the shortening of inter-spike-intervals and an increase in the firing frequency. Most importantly, spike timing was altered by carbachol perfusion. When a fluctuating current was injected, the strong spike frequency adaptation observed under a square pulse input was no longer apparent (Fig. 1B1). Unlike the results under the square pulse condition, addition of carbachol to the bath medium preserved the timing of the spikes (Fig. 1B2). An increased excitability was achieved with the insertion of additional spikes between the existing spikes.

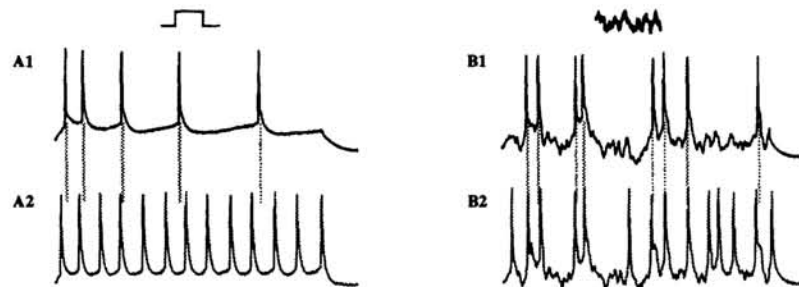

Figure 1: Response of a cortical neuron to square pulse current injection (A) and a fluctuating input (B). The membrane potential during the 1024 ms sampling period is plotted as a function of time for the two types of inputs (onset: 5 ms; duration: 900 ms). The grey lines show where the spikes occurred in the upper traces.

Preservation of spike timing under carbachol was examined at concentrations of 5, 7.5, 15, and 30$\mu$M, here shown in one cell (Fig. 2, 5$\mu$M). The smoothed histograms (as described in section 2.3) were plotted for blocks of 20 identical trials under the same fluctuating input. The alignment of the events between the control and carbachol indicates that spike timing was well preserved. The table gives the mean spike displacement, D, for a range of carbachol concentrations. The spike jitter within the control and carbachol conditions was approximately 1 ms, and was not changed significantly by carbachol (control: $0.96 \pm 0.3$; carbachol: $0.94 \pm 0.42$ ms.)

### 3.2    Simulation

The model captured the basic characteristics of experimental data. In response to fluctuating inputs, the model neurons showed reduced spike frequency adaptation and preservation of spike timing. The *in vitro* experiment were limited to only two levels of stimulus fluctuation. To show that reduced adaptation in response

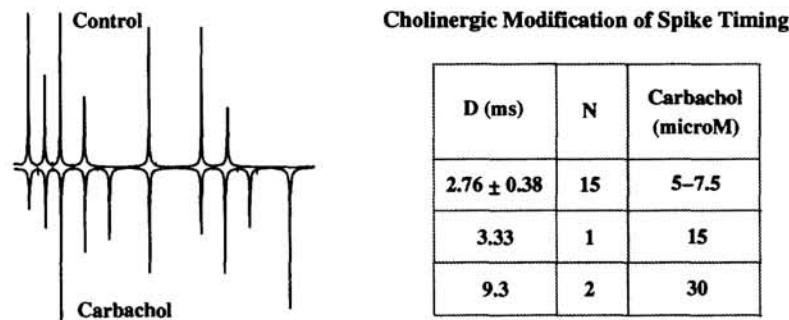

| D (ms) | N | Carbachol (microM) |
|---|---|---|
| 2.76 ± 0.38 | 15 | 5–7.5 |
| 3.33 | 1 | 15 |
| 9.3 | 2 | 30 |

Figure 2: Preservation of spike timing for a range of carbachol concentrations. Left: the top portion is the histogram for the control condition; the bottom is the histogram for the carbachol condition shown inverted. The alignment of the events between the control and carbachol indicates preserved timing. Right: statistics of spike displacement.

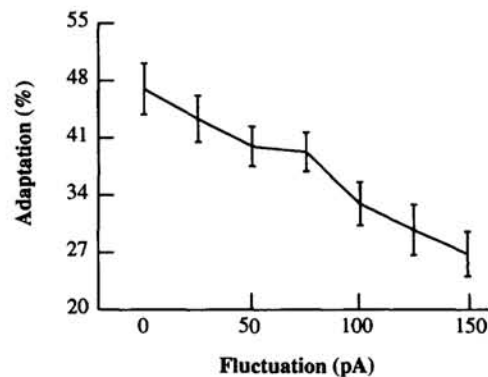

Figure 3: Reduced adaptation as a function of increasing stimulus fluctuation. Adaptation measured as a normalized spike count difference between the first and second halves of the 900 ms stimulation: (C2-C1)/C1.

to fluctuating inputs is a general phenomenon, in the model neuron we measured adaptation for multiple levels of stimulus fluctuation. As shown in Fig. 3, spike frequency adaptation decreased as a function of increasing stimulus fluctuation over a range of fluctuation amplitude. The effects cholinergic modulation on spike timing were studied under simulated cholinergic modulation. Similar to the experimental finding, increased neuronal excitability to fluctuating inputs was accompanied by insertion of additional spikes (Fig. 4 left) and spike timing was preserved simultaneously (Fig. 4 right).

In real neurons, the total effects of cholinergic modulation depends on its effects on at least three potassium conductances. Using the model, we examined the effects of manipulating each of the three potassium conductances on spike displacement and spike jitter. We found that (1) spike displacement due to reduction in potassium conductances were all very small, on the order of a few milliseconds (Fig. 5 top row); (2) Compared to the conductances underlying $I_M$ and $I_{leak}$, spike displacement was most sensitive to changes in the conductance underlying $I_{AHP}$ (Fig. 5 top row), whose reduction alone led to the best reproduction of the experimental data; (3) spike jitters of approximately 1 ms were independent of the values of the three

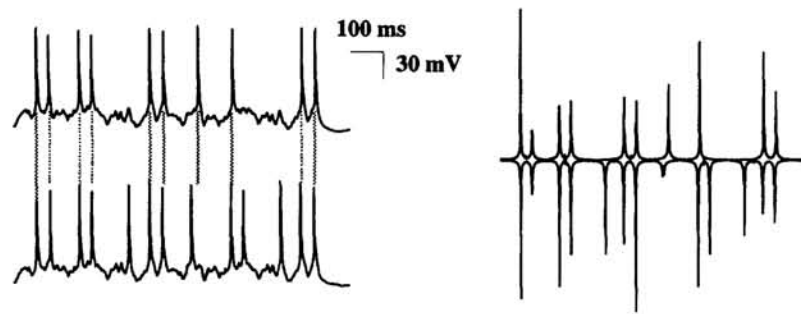

Figure 4: Preservation of spike timing in the model neocortical neuron. Left: Responses of the model neuron to fluctuating input. Top: replicating data from the control condition. Bottom: reproducing the carbachol effect by blocking the adaptation current, $I_{AHP}$. Right: histogram display of preservation of spike timing in a block of 20 trials.

potassium conductances (Fig. 5 bottom row). These results make predictions for new experiments where each individual current is blocked selectively.

## 4   Conclusions

The results showed that under the physiologically realistic fluctuating input, the effects of cholinergic modulation on spike timing are rather different from that observed when unphysiological square pulse inputs were used. Instead of moving the spikes forward in time by shortening the inter-spike-intervals, cholinergic modulation preserved spike timing. This preservation of spike timing was achieved simultaneously with an increase in neuronal excitability.

According to the classical view of neuromodulation, one would have expected that a spike timing based neural code would be difficult to maintain across a range of neuromodulator concentrations. The fact that spike timing was rather resistant to changes in the neuromodulatory environment raises the possibility that spike timing may serve some function in the cortex.

The differential effect of cholinergic modulation on spike timing observed under the square pulse and fluctuating inputs also calls for caution in generalizing an observation from one set of parameter values to another, especially when generalizing from *in vitro* to *in vivo*. This concern for external validity is particularly important for computational neuroscientists whose work involves integrating phenomena from the cellular, systems and finally, to behavioral levels.

### Acknowledgments

Supported by the Howard Hughes Medical Institute. We are grateful to Zachary Mainen, Barak Pearlmutter, Raphael Ritz, Anthony Zador, David Horn, Chuck Stevens, William Bialek, and Christof Koch for helpful discussions.

## References

Abeles, M., Bergman, H., Margalit, E., and Vaadia, E. (1993). Spatiotemporal

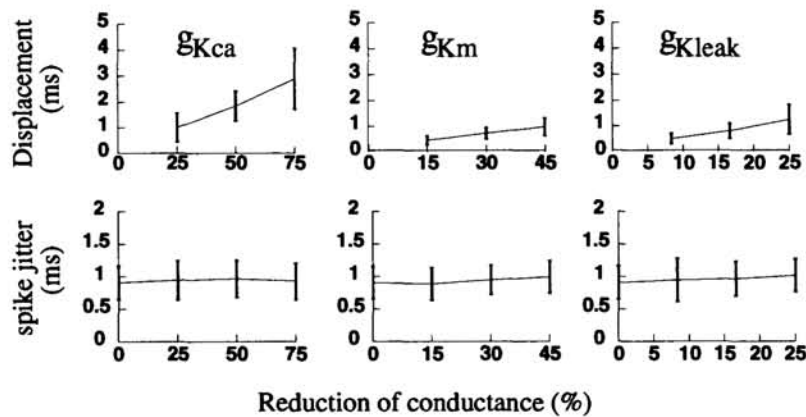

Figure 5: Effects of individual conductance changes on spike timing. Top: spike displacement as a function of changing conductances. Bottom: spike jitter as a function of changing conductances. Each conductance was reduced from its control value which was determined by fitting experimentally observed spike trains. The range of change for the leak conductance was constrained by the experimentally observed resting membrane potential changes (avg. 5 mV.)

firing patterns in the frontal cortex of behaving monkeys. *J. Neurophysiol.*, *70*, 1629–1638.

Bair, W. and Koch, C. (1996). Temporal precision of spike trains in extrastriate cortex of the behaving Macaque monkey. *Neural Computation*, *8*(6), 1184–1202.

Bialek, W. and Rieke, F. (1992). Reliability and information transmission in spiking neurons. *Trends Neurosci.*, *15*, 428–434.

Bialek, W., Rieke, F., de Ruyter van Stevenick, R. R., and Warland, D. (1991). Reading a neural code. *Science*, *252*, 1854–7.

Ferster, D. and Jagadeesh, B. (1992). EPSP-IPSP interactions in cat visual cortex studied with *in vivo* whole-cell patch recording. *J. Neurosci.*, *12*(4), 1262–1274.

Madison, D. V., Lancaster, B., and Nicoll, R. A. (1987). Voltage clamp analysis of cholinergic action in the hippocampus. *J. Neurosci.*, *7*(3), 733–741.

Mainen, Z. F. and Sejnowski, T. J. (1994). Reliability of spike timing in neocortical neurons. *Science*, *268*, 1503–6.

McCormick, D. A. (1993). Actions of acetylecholine in the cerebral cortex and thalamus and implications for function.. *Prog. Brain Res.*, *98*, 303–308.

Nicoll, R. (1988). The coupling of neurotransmitter receptors to ion channels in the brain. *Science*, *241*, 545–550.

Rieke, F., Warland, D., de Ruyter van Steveninck, R., and Bialek, W. (1996). *Spikes: Exploring the Neural Code*. MIT Press.

Shadlen, M. N. and Newsome, W. T. (1994). Noise, neural codes and cortical organization. *Current Opinion in Neurobiology*, *4*, 569–579.

Stevens, C. and Zador, A. (1995). The enigma of the brain. *Current Biology*, *5*, 1–2.